# Feature Transitions with Saccadic Search: Size, Color, and Orientation Are Not Alike

**Stella X. Yu**
Computer Science Department
Boston College
Chestnut Hill, MA 02467
`stella.yu@bc.edu`

## Abstract

Size, color, and orientation have long been considered elementary features whose attributes are extracted in parallel and available to guide the deployment of attention. If each is processed in the same fashion with simply a different set of local detectors, one would expect similar search behaviours on localizing an equivalent flickering change among identically laid out disks. We analyze feature transitions associated with saccadic search and find out that size, color, and orientation are not alike in dynamic attribute processing over time. The Markovian feature transition is attractive for size, repulsive for color, and largely reversible for orientation.

## 1 Introduction

Size, color, and orientation have long been considered elementary features [14] that are available to guide attention and visual search [17]. Their special status in early visual processing is supported by a large volume of psychophysical evidence on how they can mediate effortless texture segregation, recombine in illusory conjunctions, and pop out in feature search [13, 16]. There is also physiological evidence on how these features could be extracted with separate sets of dedicated detectors working in parallel across the entire space [6]. Consequently, in schematic diagrams as well as computational models on visual saliency [13, 3, 12], image segmentation [5], object recognition [10, 4, 20], and scene classification [12, 11], it is routinely assumed that features at all scales, colors, and orientations are processed and available simultaneously.

While size, color, and orientation are alike at parallel local detections across space, they may not be alike at serial deployment of attention across time. We investigate this issue in a gaze-tracked visual search experiment which often requires multiple saccades for the subject to locate the target (Fig. 1).

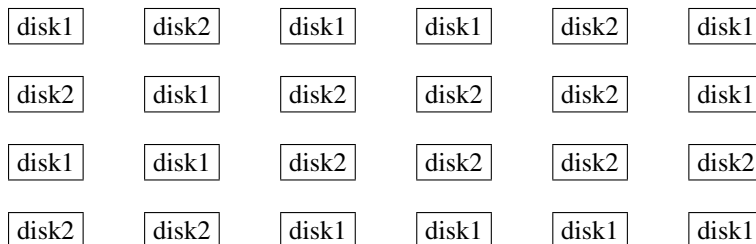

Figure 1: Two kinds of disks are uniformly randomly distributed in a fixed regular layout. Only one disk changes its kind during a repeated flickering presentation. For the same size of change, does it matter to visual search whether the two kinds of disks are rendered in size, color, or orientation?

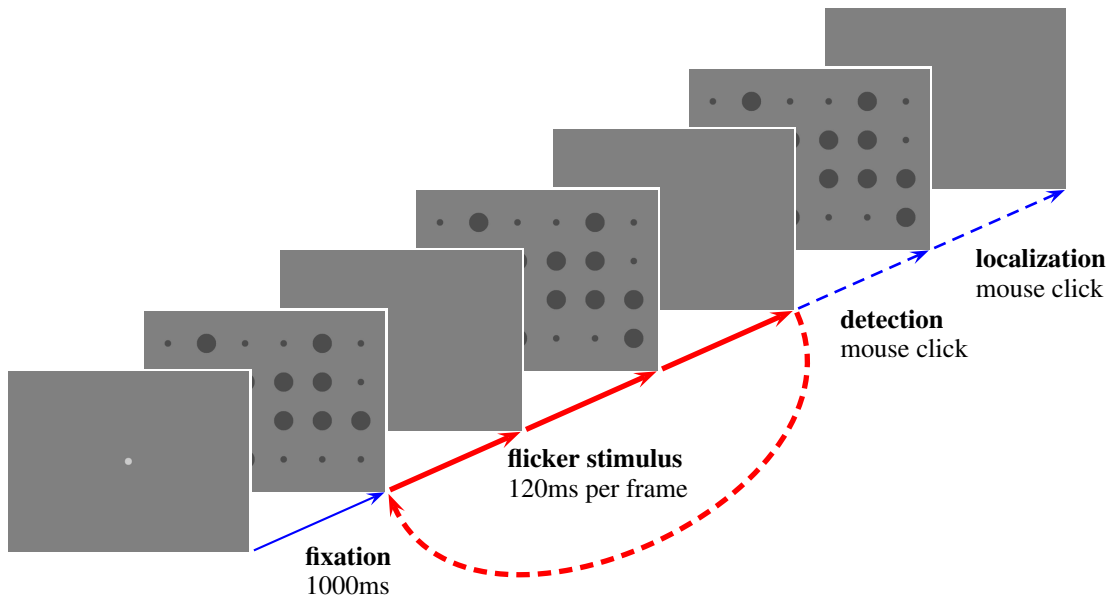

Figure 2: Each trial goes through fixation, stimulus, detection, and localization stages. A fixation dot is displayed for 1 second before the onset of the flicker stimulus, with disk image 1, blank, disk image 2, blank repeatedly presented for 120ms each. The subject issues a mouse click as soon as he detects the change, and the the last seen disk image remains on till he clicks the disk of change. A blank screen is then displayed for 2 seconds before the start of next trial.

We present two kinds of disks in a fixed regular layout in a flicker paradigm, and the subject's task is to locate the only disk that changes its kind (Fig. 2). The paradigm induces *change blindness*, where a large difference between two images becomes strikingly difficult to detect with a blank in-between, even with repeated presentations [9, 2, 8, 18] . Without the blank, the change elicits a singular motion signal which automatically draws the viewer's attention to the location of change; with the blank, the motion signal is disrupted and overwhelmed by those motion transients between either image and the blank, effectively masking the location of change.

If the magnitude of change is comparable across feature dimensions, does it matter whether the disks are rendered in size, color, or orientation? That is, does visual search vary according to whether the same array of disks are: 1) small and large, 2) black and white, or 3) horizontal and vertical disks? If size, color, and orientation are processed in the same fashion with dedicated local detectors operating in parallel across space, then the detector responses are identical spatially at any time instance in the 3 scenarios. The question is whether the deployment of attention, i.e. deciding what disks to look at next and how to look, depend on which filters produce these responses.

Note that our stimuli decouple the target of feature search from visual saliency in the space. Our target is defined not by one of the attributes as done in static search displays [14, 17], but by the temporal change of the attribute. At any time instance, the attributes are uniformly random everywhere, so the target cannot draw attention to itself, but has to be discovered with search. The effect of the attribute itself on attention can thus be studied without the confounding factor of saliency.

The focus of this paper is on how the feature space is navigated with saccadic search. We formulate a feature descriptor for each fixation, based on which we develop a Markovian feature transition model for saccadic eye movements. Our model reveals that feature transition is attractive for size, repulsive for color, and largely reversible for orientation, suggesting that size, color, and orientation are not alike in dynamic attribute processing over time.

## 2 Gaze-Tracked Change Blindness Experiment

We investigate whether visual search for attribute change differs when the stimulus is rendered in size, color, or orientation with the same layout. We establish in a separate experiment that the change is equivalent among dimensions: Detection is equally fast and accurate for a change between two attributes across dimensions and for a no-change within each dimension across two attributes.

| flicker stimuli | size | color | orientation |
|---|---|---|---|
| identical spatial layout |  <br> 1    2 |  <br> 1    2 |  <br> 1    2 |
| 1st image <br> 1 2 1 1 2 1 <br> 2 1 2 2 2 1 <br> 1 1 1 2 2 2 <br> 2 2 2 1 1 ①  | 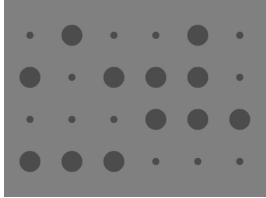 | 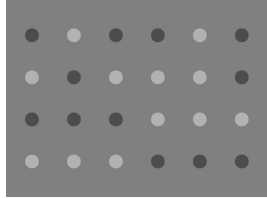 | 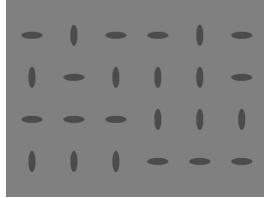 |
| 2nd image <br> 1 2 1 1 2 1 <br> 2 1 2 2 2 1 <br> 1 1 1 2 2 2 <br> 2 2 2 1 1 ②  | 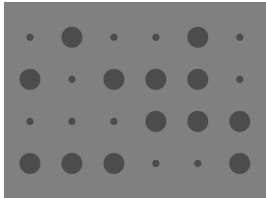 | 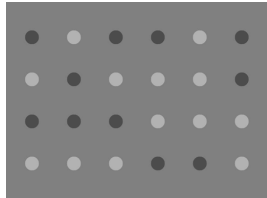 | 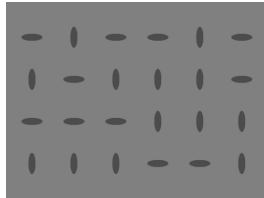 |

Figure 3: Flicker stimuli are rendered in the same layout but separately in size, color, and orientation. The 1st image contains 12 attribute-1 disks and 12 attribute-2 disks in a uniformly random spatial distribution. The 2nd image is identical to the 1st image except that 1 disk changes its attribute. It could be any of the 24 disks. The disk of change here is circled in both layout matrices.

**Stimuli.** There are 2 kinds of disks for each dimension. Size has 2 radii, $0.45°$ for small and $1.35°$ for large. Color has 2 values, $0.3$ for black and $0.7$ for white on $0-1$ value scale. Orientation has 2 angles, $0°$ for horizontal and $90°$ for vertical, with disk radii $0.45°\times1.35°$ along two directions. Both size and orientation stimuli are of black value $0.3$. Color stimuli are of medium disk radius $0.9°$. The background is of neutral gray value $0.5$. Here we restrict color to luminance only, as color hue processing is uniquely foveal, which would greatly confound explanations for search behaviours.

The flicker stimuli for the 3 dimensions are rendered in an identical spatial layout. Each stimulus involves a pair of 24-disk images which are identical except for one disk. These 24 disks are located centrally on a regular $4 \times 6$ grid, with an inter-disk distance of $5.4°$, which is $4$ times the maximal radius a disk could assume. The 1st image of the stimulus consists of uniformly randomly distributed 12 attribute-1 disks and 12 attribute-2 disks. The 2nd image changes one of the 24 disks (Fig. 3).

**Apparatus.** The display extends $25.6° \times 34.1°$ at a viewing distance of 5 meters. Gaze data are recorded with a Tobii x50 eye tracker at 50Hz sampling rate and $0.5°$-$0.7°$ accuracy. Two clock-synced 3.2GHz Dell Precision computers control the eye tracker and the stimulus presentation respectively. The eye tracker is calibrated at the beginning of each data recording session.

**Procedure.** Each trial begins with a fixation dot of radius $0.5°$ shown at the center of the display for 1 second. The flicker stimulus, in the sequence of disk image 1, blank, disk image 2, and blank, is then repeatedly presented for 120 ms each. Once the subject issues a mouse click to indicate his detection of the change, the last disk image remains on till the location of change is clicked (Fig. 2).

There are 3 sets of random stimuli run in 3 sessions. Each session has 3 blocks of 24 trials each, one trial for one change location and one block for one dimension. The trials are completely randomized in a block, and the blocks are also randomized and balanced among the subjects.

The subject is told that two images differing in only one disk are presented repeatedly. His task is to detect and localize the changing disk. He should issue a click as soon as he detects the change. The flickering then stops at the last seen disk image, and he should click the disk of change.

**Participants.** A total of 24 naive subjects with normal or corrected-to-normal vision participated after providing informed consent and were compensated with cash. 11, 8, and 5 subjects took part in one, two and all three sessions respectively.

# 3 Performance Analysis

We evaluate the task performance on both the accuracy measured by the percentage of correct change localizations and the reaction time measured from the flicker stimulus onset to the subject's first mouse click for indicating a detection. Fig. 4 shows that localizing an equivalent change among identically laid out items yields significantly different performances in the 3 dimensions. It is fastest and most accurate in size, less so in orientation, and least in color.

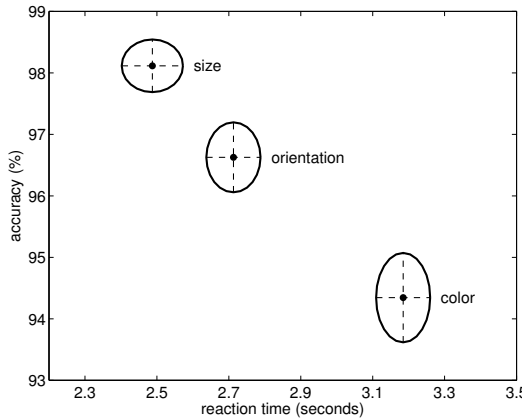

Figure 4: Change localization given an identical layout is best (fastest and most accurate) in size, worse in orientation, and worst in color. The sample means and their standard errors of reaction times ($x$-axis) and accuracies ($y$-axis) are indicated by the centers and radii of ellipses respectively. The differences are significant, with one-way ANOVA results of $F(2, 3021) = 10.43$, $p = 3.1 \times 10^{-5}$ for accuracy and $F(2, 3021) = 20.43$, $p = 1.5 \times 10^{-9}$ for reaction time. We treat the data from all the subjects as samples from a single subject population, since we are interested not in individual subjects' performance, but in the distinction between feature dimensions.

The human visual system must accomplish change localization by examining more than one disk per flicker cycle, since the mean reaction time is only about 5, 6, and 7 cycles ($0.12 \times 4 = 0.48$ seconds per cycle) for size, orientation, and color respectively. If only one item is looked at and ruled out per cycle, on average it would require fixating 50% of 24 disks till hitting the target disk, i.e. in 12 flicker cycles. Our average of 6 cycles suggests that about 2 disks are examined per cycle.

When a disk is being fixated, all its 8 neighbouring disks are mostly out of fovea, since they are either $5.4°$ or $7.7°$ apart. Some coarse information about neighbouring disks must be utilized in each fixation. The neighbourhood effect on change localization is studied in Fig. 5 and Fig. 6.

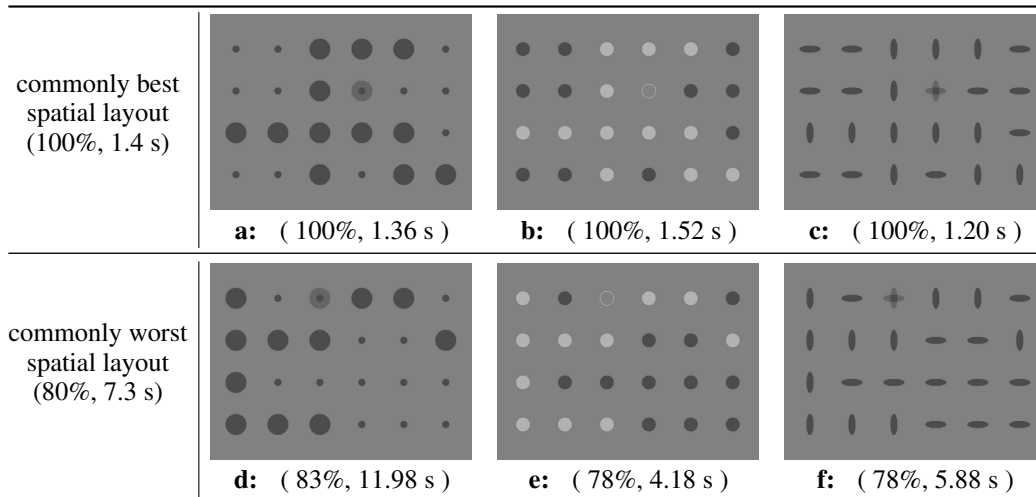

Figure 5: The common spatial layout that yields the best (**a,b,c**) or the worst (**d,e,f**) change localization performance in all 3 feature dimensions. Each pair of numbers ($a$%, $b$ s) indicate mean accuracy $a$ and reaction time $b$. Shown here is the average image of a flicker stimulus, with the disk of change taking two attributes, except in the case of color: Since the average has the same intensity as background, the change is outlined in white instead. The commonly best layout has the change among uniform attributes, whereas the commonly worst layout has a mixture of attributes.

| dimension | **size** | **color** | **orientation** |
|---|---|---|---|
| dimension-<br>specific<br>best<br>spatial layout | 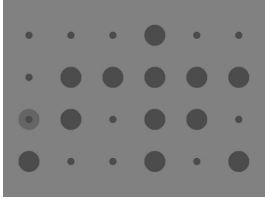 | 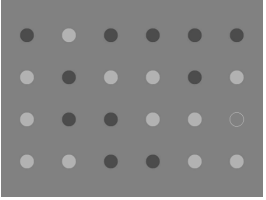 | 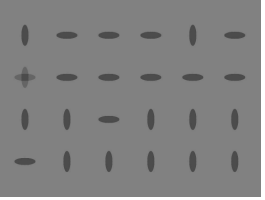 |
| | **a:** ( **100%, 2.34 s** )<br>( 83%, 5.44 s )<br>( 83%, 6.47 s ) | **b:** ( 100%, 3.08 s )<br>( **100%, 2.09 s** )<br>( 80%, 2.35 s ) | **c:** ( 94%, 3.37 s )<br>( 78%, 4.08 s )<br>( 94%, **2.69 s** ) |
| dimension-<br>specific<br>worst<br>spatial layout | 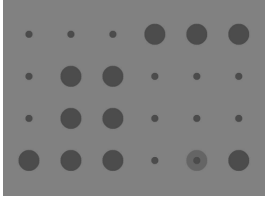 | 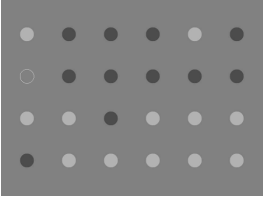 | 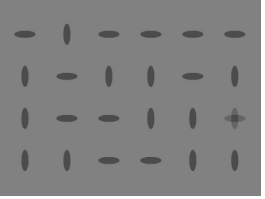 |
| | **d:** ( **90%,** 2.10 s )<br>( 100%, 2.65 s )<br>( 100%, 2.59 s ) | **e:** ( 94%, 3.37 s )<br>( **78%, 4.08 s** )<br>( 94%, 2.69 s ) | **f:** ( 100%, 3.08 s )<br>( 100%, 2.09 s )<br>( **80%, 2.35 s** ) |

Figure 6: The dimension-specific spatial layout that yields the best or worst change localization performance in one dimension only, with the largest performance gap over the other 2 dimensions. Same convention as Fig. 5. The 3 rows of numbers below each image indicate the mean accuracy and reaction time for a stimulus rendered in the same layout but in size, color, and orientation respectively. The localization of a flickering change is easier in a primarily large neighbourhood for size, in any homogeneous neighbourhood for color, and in a collinear neighbourhood for orientation.

Fig. 5 shows that a uniform neighbourhood tends to facilitate change localization, whereas a mixed neighbourhood tends to hinder change localization, no matter which dimension the disks are rendered in. Fig. 6 shows distinctions in the neighbourhood uniformity between the 3 dimensions.

**For size**, change localization is easier in a neighbourhood populated with large disks. If the dominant size is large (Fig. 6a), missing a large would be easier to detect, whereas if the dominant size is small (Fig. 6d), missing a small would be difficult to detect. That is, unlike color or orientation, the attributes of size are asymmetrical: small produces a smaller response than large, with size 0 for a response of 0 in the limiting case. When neither small nor large is dominant in the neighbourhood (Fig. 5d), change localization becomes most difficult. **For color**, change localization is easier if one color, either black or white, dominates the neighbourhood. **For orientation**, it is easier only if the oriented disk is part of collinear layout.

## 4   Feature Analysis with Eye Movements

Having seen that neighbourhood uniformity has an impact on the change localization performance, we investigate how it influences the decision on which item to look at in the next fixation.

We first associate a fixation with a set of $f$-numbers at that location, each measuring the overall attribute density in a neighbourhood defined by a Gaussian spatial weighting function. Let $\mathrm{loc}(i)$ denote the location of pixel $i$, $\mathrm{dist}(i,j)$ the distance between pixels $i$ and $j$, and $G(x;\sigma)$ the 1D Gaussian function of $x$ with mean 0 and standard deviation $\sigma$. We have:

$$f_0(i) = \begin{cases} 0, & \text{no disk at } \mathrm{loc}(i) \\ -1, & \text{disk type 1 at } \mathrm{loc}(i) \\ 1, & \text{disk type 2 at } \mathrm{loc}(i) \end{cases} \tag{1}$$

$$f_\sigma(i) = \frac{\sum_j f_0(j) G(\mathrm{dist}(i,j);\sigma)}{\sum_j G(\mathrm{dist}(i,j);\sigma)} \tag{2}$$

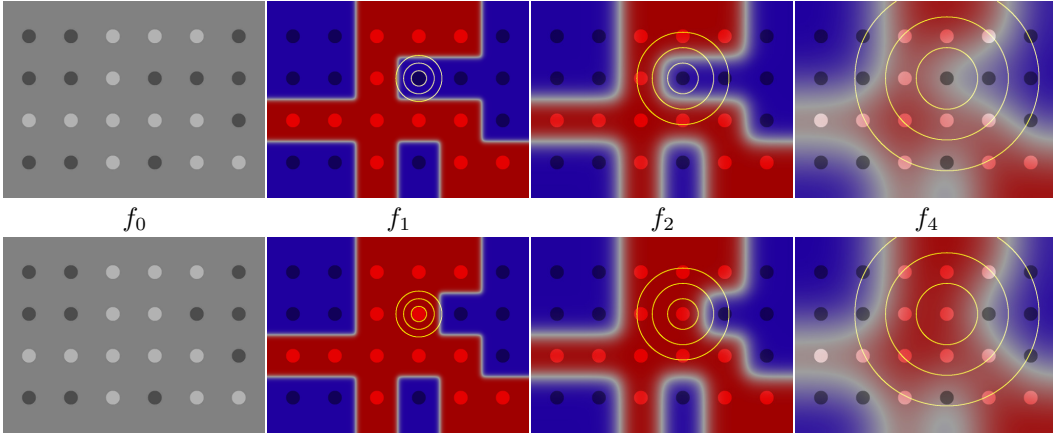

Figure 7: The $f$-number images of a flicker stimulus. A negative $f$ number (in blue shades) indicates the dominance of attribute 1, whereas a positive $f$ number (in red shades) indicates the dominance of attribute 2. The closer the $f$ number is to 0 (in gray shades), neither attribute dominates the neighbourhood. $f_\sigma$ measures the average attribute in a Gaussian neighbourhood with standard deviation $\sigma$. The 3 circles on the target of change mark the $\sigma, 2\sigma, 3\sigma$ radii. While $\sigma = 1$ covers only one disk in isolation, $\sigma = 2$ also covers 8 adjacent disks, and $\sigma = 4$ covers 16 adjacent disks.

An $f$ value of $-1, 1, 0$ indicates the dominance of attribute 1, 2 or neither. With an increasing $\sigma$, $f_\sigma$ estimates the majority of attributes in a larger neighbourhood.

Each location is now associated with a set of $f$ numbers, $(f_1, f_2, \ldots)$, and they as a whole capture the attribute homogeneity surrounding that location. Fig. 7 shows $f$ for the best spatial layout in Fig. 5. At $\sigma = 1$, the neighbourhood could only contain one disk, thus $f_1(i) = f_0(i)$ for most location $i$'s. At $\sigma = 2$, it also contains 8 adjacent neighbours: $f_2(i) = f_1(i)$ for $i$ in a uniform neighbourhood, and $f_2(i) \approx 0$ for $i$ in a mixed neighbourhood. At $\sigma = 4$, the neighbourhood is about the half size of the display, with $f_4(i) = 0$ for $i$ bordering two large different uniform neighbourhoods.

Fig. 8 shows the distributions of $f$ associated with all the fixations. The two peaks of $f_1$ in all the 3 feature dimensions demonstrate that visual search tends to fixate disks rather than empty spaces between disks. There is also an attribute bias in each dimension, and the bias is weakest in orientation and strongest in size. This bias is not diminished in $f_2$, demonstrating that visual search tends to navigate in groups of large disks. The single peak of $f_4$ at value 0 not only confirms the uniform randomness of our stimuli, but also reveals that the empty spaces being fixated tend to be those borders between different attribute neighbourhoods at a coarser scale (Fig. 7 Column 4).

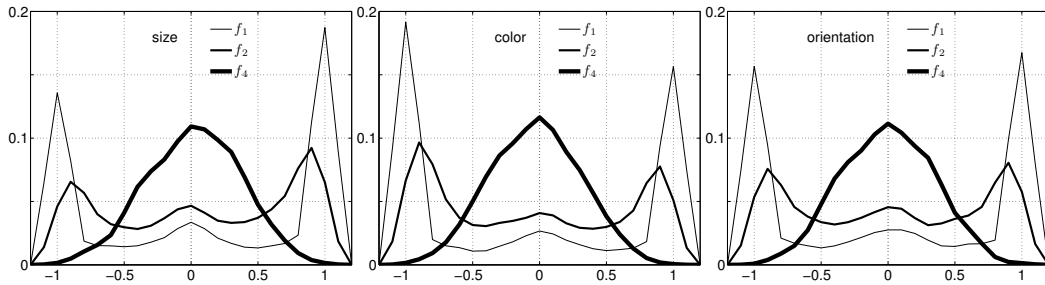

Figure 8: The probability distribution of $f_1, f_2, f_4$ (in increasing line widths) associated with all the fixations shows a strong preference in size for large disks as well as areas of large disks ($+1$), a small preference in color for black disks ($-1$), and a slight preference in orientation for vertical disks ($+1$). The single peak of $f_4$ at 0 reveals most fixations occurring near those disks separating large groups of uniform attributes. These statistics are robust with respect to subject sub-sampling validation, e.g. over 8 samplings of 10 subjects only, the maximal standard error is 0.006.

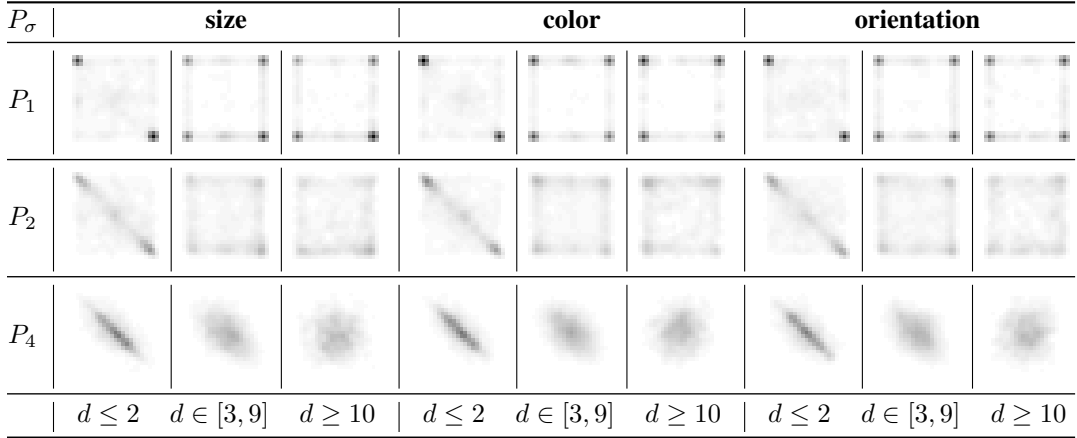

| $P_\sigma$ | size | | | color | | | orientation | | |
|---|---|---|---|---|---|---|---|---|---|
| $P_1$ | | | | | | | | | |
| $P_2$ | | | | | | | | | |
| $P_4$ | | | | | | | | | |
| | $d \leq 2$ | $d \in [3,9]$ | $d \geq 10$ | $d \leq 2$ | $d \in [3,9]$ | $d \geq 10$ | $d \leq 2$ | $d \in [3,9]$ | $d \geq 10$ |

Figure 9: The probability distribution of $f_1, f_2, f_4$ associated with all the saccades shows a preference of jumping to a disk of the same attribute regardless of saccade distance $d$ and neighbourhood size $\sigma$. Each transition $P(a, b; d, \sigma)$ given $d$ and $\sigma$ is visualized as a 2D image, e.g. for size, the right lower corner of $P_1$ is the frequency of saccading from large to large. A darker gray indicates a larger transitional probability. As $d$ increases, it is more likely to jump to a different attribute, and the chance is more uniformly random in orientation.

Fig. 9 shows the joint distributions of two $f$ numbers associated with the initiating fixations and the landing fixations of all the saccades, organized according to the saccade distance. For a saccade from pixel $i$ to pixel $j$, it contributes one count of transition from $a$ to $b$ in the $f$-space:

$$P_\sigma(a, b|d) = \text{Prob}(f_\sigma(i) = a, f_\sigma(j) = b, \text{loc}(i) \xrightarrow{\text{saccade}} \text{loc}(j)| \text{dist}(i, j) = d) \tag{3}$$

Fig. 9 shows that all the transitions within $2°$ tend to cluster tightly along the diagonals, i.e. between the same attributes. At such a short distance, each saccade could not reach a different disk. These transitions are thus between the same disks or between the same inter-disk empty spaces, by e.g. micro-saccades. The bias towards a particular attribute is also clear in each dimension: There are more transitions between larges than between smalls, more between blacks than between whites, about the same between horizontals and between verticals. As the saccade distance increases, disks of various attributes become viable candidates to saccade to. It becomes more likely to saccade to another disk of the same or different attribute than to saccade to an empty space (i.e. low probabilities in the middle rows or columns of $P_\sigma$ images).

We further examine $P_\sigma(a, b|d)$ at $\sigma = 1$ and $d$ in the middle range of $[2°, 10°]$, associated with saccades towards adjacent disks. We quantize $f$ into 3 values based on a threshold $\theta$: $-1$ if $f < -\theta$, $0$ if $|f| < \theta$, and $1$ if $f > \theta$. The joint probability can be decomposed into marginal probability $\pi(a)$ at the initiating attribute $a$ and conditional probability $P(b|a)$ for the landing attribute $b$:

$$P(a, b) = \pi(a) \times P(b|a) = \left( \sum_c P(a, c) \right) \times \left( \frac{P(a, b)}{\sum_c P(a, c)} \right) \tag{4}$$

While $\pi(a)$ measures the proportion of fixations at attribute $a$ among all the fixations, $P(b|a)$ measures the proportion of saccades towards $b$ given the current fixation at $a$. Consistent with Fig. 8, $\pi(a)$ in Fig. 10 shows more visits to large, black, vertical than to small, white, and horizontal.

The most interesting finding comes from $P(b|a)$: While the attributes are uniformly random in the neighbourhood, our eyes do not act like a blind space wanderer. **1) For size**, it is much more likely to visit large no matter what is being looked at in the current fixation, i.e., attribute large is an attractor in the $f$ space. **2) For color**, it is more likely to visit black from either black or white, but not from an empty space, i.e., if no disk is in fixation, it is more likely to visit white in the next fixation. Unlike size, white is not an attractor, but a repeller: Once at white, the eyes are more inclined to leave for black than staying in the group of whites. **3) For orientation**, it is only slightly more likely to visit vertical than horizontal. When the eyes are on an empty space, it is in fact equally likely to visit horizontal or vertical in the next fixation, i.e., there is no strong attractor or repeller in orientation, and the two attributes are largely reversible. Such biases also persist over larger saccades.

| **size** | | | | | **color** | | | | | **orientation** | | | |
|---|---|---|---|---|---|---|---|---|---|---|---|---|---|
| $\pi(a)$ | $P(b\|a)$ |  | |  | $\pi(a)$ | $P(b\|a)$ |  | |  | $\pi(a)$ | $P(b\|a)$ | 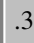 | |  |
| .43 |  | .40 | .10 | .50 | .52 |  | .53 | .09 | .38 | .46 |  | .44 | .08 | .48 |
| .04 | | .36 | .15 | .49 | .04 | | .33 | .16 | .51 | .04 | | .44 | .12 | .44 |
| .53 |  | .37 | .12 | .51 | .44 |  | .48 | .09 | .43 | .50 |  | .43 | .10 | .47 |

Figure 10: The probability distribution of $f_1$ for all the saccades within $[2°, 10°]$. $f_1$ is quantized into $-1, 0, 1$, corresponding to attribute 1, empty space, and attribute 2 respectively, based on threshold $\theta = 0.15$. These statistics are validated over 13 leave-50%-subjects-out samplings, with the standard error for each number less than $0.01$ except for the second row of $P(b|a)$(valued at $0.02, 0.01, 0.02$ instead). Let $a$ and $b$ denote attributes, or row and column indices into the transition table. $\pi(a)$ is the overall probability of looking at $a$. $P(b|a)$ is the probability of saccading to $b$ at $a$. For example, for size, $\pi$ shows that $43\%$ of all the fixations look at small, $4\%$ at empty, and $53\%$ at large, whereas the 3rd row of $P$ shows that upon fixating at large, there is $51\%$ chance of saccading to another large, $37\%$ chance to a small disk and $12\%$ chance to an empty space. The most likely action is highlighted in red. Search in size tends to be attracted to large, search in color tends to be repelled by white, whereas search in orientation is largely reversible between horizontal and vertical.

These results cannot be explained by visual crowding, where the perception of peripherally viewed shapes is impaired with nearby similar shapes [7]. While critical spacing is always roughly half the viewing eccentricity and independent of stimulus size, crowding magnitude differs across features: Size crowding is almost as strong as orientation crowding, whereas the effect is much weaker for color [15]. Therefore, feature crowding cannot explain the different natures of feature transitions for size, color, and orientation, or why such biases persist over larger saccades.

# 5 Summary

Size, color, and orientation are considered elementary features extracted with separate sets of detectors responding in parallel across space. They are modeled by the same computational mechanism, differing only in the filters that implement their local attribute detectors.

We conducted a gaze-tracked change blindness experiment, where the subject needs to locate a flickering change among items rendered identically in space and separately in size, color, and orientation. If the deployment of attention during search depends only on the master spatial map of responses [14, 13, 3, 17, 12], regardless of which type of filters produces them, we should observe little differences in the search performance and behaviours among the 3 dimensions.

Our search performance analysis shows that change localization is fastest and most accurate in size, less in orientation, worst in color. Change in a uniform neighbourhood is easier to localize, but only if the attribute is large for size, or if the items form collinear extension for orientation.

Our feature analysis with eye movements shows that search in each dimension has an attribute bias: large for size, black for color, and vertical for orientation, and a common spatial bias on border items separating large uniform groups. However, feature transitions with saccades have a strong attractor bias for large, and a repeller bias for white, and a very little bias for orientation.

These biases create an interesting dynamics in serial processing over time which could explain why localization is most effective in size and worst in color. It is not due to their alike local detectors in the space, but due to their own selectivity in grouping [8, 19, 1] over time: Focusing on the large group essentially cuts down the search space by half, whereas excursion into the white group from the primary black group only hurts the spatial efficiency of search.

Our results and analysis methods on these elementary features thus provide new insights into the computation of visual saliency and task-specific visual features across dimensions and over time.

## Acknowledgements

This research is funded by NSF CAREER IIS-0644204 and a Clare Boothe Luce Professorship. I would like to thank Dimitri Lisin, Marcus Woods, Sebastian Skardal, Peter Sempolinski, David Tolioupov, and Kyle Tierney for earlier discussions and excellent assistance with the experiments. I am grateful for many insightful comments I have received from Jeremy Wolfe, Ronald Rensink, and anonymous reviewers; their valuable suggestions have greatly improved the paper.

## References

[1] G. Fuggetta, S. Lanfranchi, and G. Campana. Attention has memory: priming for the size of the attentional focus. *Spatial Vision*, 22(2):147–59, 2009.

[2] J. Grimes. On the failure ot detect changes in scenes across saccades. 2, 1996.

[3] L. Itti and C. Koch. Computational modelling of visual attention. *Nature Neuroscience*, pages 194–203, 2001.

[4] D. G. Lowe. Distinctive image features from scale-invariant keypoints. 2003.

[5] J. Malik, S. Belongie, T. Leung, and J. Shi. Contour and texture analysis for image segmentation. *International Journal of Computer Vision*, 2001.

[6] J. H. R. Maunsell and W. T. Newsome. Visual processing in monkey extrastriate cortex. *Annual Review of Neuroscience*, 10:363–401, 1987.

[7] D. G. Pelli, M. Palomares, and N. J. Majaj. Crowding is unlike ordinary masking: distinguishing feature integration from detection. *Journal of Vision*, 4(12):1136–69, 2004.

[8] R. Rensink. Visual search for change: A probe into the nature of attentional processing. *Visual Cognition*, 7:345–76, 2000.

[9] R. A. Rensink, J. K. O'Regan, and J. J. Clark. Image flicker is as good as saccades in making large scene changes invisible. *24*, pages 26–8, 1995.

[10] M. Riesenhuber and T. Poggio. Hierarchical models of object recognition in cortex. *Nature Neuroscience*, 2(11):1019–25, 1999.

[11] T. Serre, A. Oliva, and T. Poggio. A feedforward architecture accounts for rapid categorization. *Proceedings of National Academy of Sciences*, 104(15):6424–9, 2007.

[12] A. Torralba. Contextual influences on saliency. In L. Itti, G. Rees, and J. Tsotsos, editors, *Neurobiology of Attention*, pages 586–93. Academic Press, 2004.

[13] A. Treisman. The perception of features and objects. In R. D. Wright, editor, *Visual Attention*. Oxford University Press, 1998.

[14] A. Treisman and G. Gelade. A feature-integration theory of atttention. *Cognitive Psychology*, 12(1):97–136, 1980.

[15] R. van den Berg, J. B. T. M. Roerdink, and F. W. Cornelissen. On the generality of crowding: visual crowding in size, saturation, and hue compared to orientation. *Journal of Vision*, 7(2):1–11, 2007.

[16] J. M. Wolfe. Asymmetries in visual search: an introduction. *Perception and Psychophysics*, 63:381–9, 2001.

[17] J. M. Wolfe and T. S. Horowitz. What attributes guide the deployment of visual attention and how do they do it? *Nature Neuroscience*, 5, 2004.

[18] J. M. Wolfe, A. Reinecke, and P. Brawn. Why don't we see changes? the role of attentional bottlenecks and limited visual memory. *Visual Cognition*, 19(4-8):749–80, 2006.

[19] Y. Yeshurun and M. Carrasco. The effects of transient attention on spatial resolution and the size of the attentional cue. *Perception and Psychophysics*, 70(1):104–13, 2008.

[20] H. Zhang, A. C. Berg, M. Maire, and J. Malik. SVM-KNN: Discriminative nearest neighbor classification for visual category recognition. In *IEEE Conference on Computer Vision and Pattern Recognition*, pages 2126–36, 2006.

